# Semantic Kernel Forests from Multiple Taxonomies

**Sung Ju Hwang**
University of Texas
Austin, TX 78701
sjhwang@cs.utexas.edu

**Kristen Grauman**
University of Texas
Austin, TX 78701
grauman@cs.utexas.edu

**Fei Sha**
University of Southern California
Los Angeles, CA 90089
feisha@usc.edu

## Abstract

When learning features for complex visual recognition problems, labeled image exemplars alone can be insufficient. While an *object taxonomy* specifying the categories' semantic relationships could bolster the learning process, not all relationships are relevant to a given visual classification task, nor does a single taxonomy capture all ties that *are* relevant. In light of these issues, we propose a discriminative feature learning approach that leverages *multiple* hierarchical taxonomies representing different semantic views of the object categories (e.g., for animal classes, one taxonomy could reflect their phylogenic ties, while another could reflect their habitats). For each taxonomy, we first learn a tree of semantic kernels, where each node has a Mahalanobis kernel optimized to distinguish between the classes in its children nodes. Then, using the resulting *semantic kernel forest*, we learn class-specific kernel combinations to select only those relationships relevant to recognize each object class. To learn the weights, we introduce a novel hierarchical regularization term that further exploits the taxonomies' structure. We demonstrate our method on challenging object recognition datasets, and show that interleaving multiple taxonomic views yields significant accuracy improvements.

## 1 Introduction

Object recognition research has made impressive gains in recent years, with particular success in using discriminative learning algorithms to train classifiers tuned to each category of interest (e.g., [1, 2]). As the basic "image features + labels + classifier" paradigm has reached a level of maturity, we believe it is time to reach beyond it towards models that incorporate richer *semantic* knowledge about the object categories themselves.

One appealing source of such external knowledge is a taxonomy. A hierarchical semantic taxonomy is a tree that groups classes together in its nodes according to some human-designed merging or splitting criterion. For example, well-known taxonomies include WordNet, which groups words into sets of cognitive synonyms and their super-subordinate relations [3], and the phylogenetic tree of life, which groups biological species based on their physical or genetic properties. Critically, such trees implicitly embed cues about human perception of categories, how they relate to one another, and how those relationships vary at different granularities. Thus, in the context of visual object recognition, such a structure has the potential to guide the selection of meaningful low-level features, essentially augmenting the standard supervision provided by image labels. Some initial steps have been made based on this intuition, typically by leveraging the WordNet hierarchy as a prior on inter-class visual similarity [4, 5, 6, 7, 8, 9, 10, 11].

Two fundamental issues, however, complicate the use of a semantic taxonomy for learning visual objects. First, a given taxonomy may offer hints about visual relatedness, but its structure need not entirely align with useful splits for recognition. (For example, *monkey* and *dog* are fairly distant semantically according to WordNet, yet they share a number of visual features. An *apple* and *apple-sauce* are semantically close, yet are easily separable with basic visual features.) Second, given the complexity of visual objects, it is highly unlikely that some *single* optimal semantic taxonomy exists to lend insight for recognition. While previous work relies on a single taxonomy out of convenience,

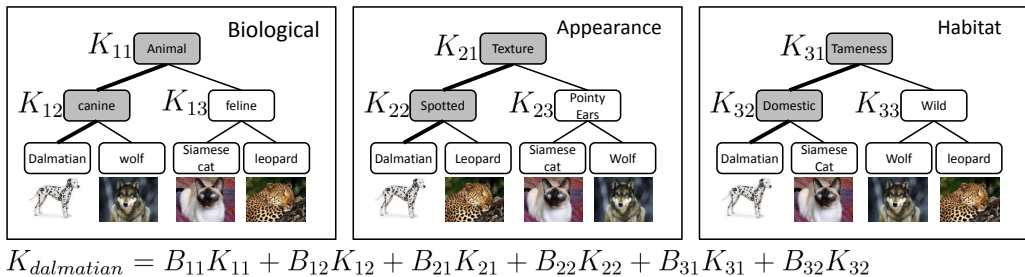

$$K_{dalmatian} = B_{11}K_{11} + B_{12}K_{12} + B_{21}K_{21} + B_{22}K_{22} + B_{31}K_{31} + B_{32}K_{32}$$

Figure 1: **Main idea:** For a given set of classes, we assume multiple semantic taxonomies exist, each one representing a different "view" of the inter-class semantic relationships. Rather than commit to a single taxonomy—which may or may not align well with discriminative visual features—we learn a tree of kernels for each taxonomy that captures the granularity-specific similarity at each node. Then we show how to exploit the inter-taxonomic structure when learning a combination of these kernels from multiple taxonomies (i.e., a "kernel forest") to best serve the object recognition tasks.

in reality objects can be organized along many semantic dimensions or "views". (For example, a *Dalmatian* belongs to the same group as the *wolf* according to a biological taxonomy, as both are canines. However, in terms of visual attributes, it can be grouped with the *leopard*, as both are spotted; in terms of habitat, it can be grouped with the *Siamese cat*, as both are domestic. See Figure 1.)

Motivated by these issues, we present a discriminative feature learning approach that leverages *multiple* taxonomies capturing different semantic views of the object categories. Our key insight is that some combination of the semantic views will be most informative to distinguish a given visual category. Continuing with the sketch in Figure 1, that might mean that the first taxonomy helps learn dog- and cat-like features, while the second taxonomy helps elucidate spots and pointy corner features, while the last reveals context cues such as proximity to humans or indoor scene features. While each view differs in its implicit human-designed splitting criterion, all separate some classes from others, thereby lending (often complementary) discriminative cues. Thus, rather than commit to a single representation, we aim to inject pieces of the various taxonomies as needed.

To this end, we propose *semantic kernel forests*. Our method takes as input training images labeled according to their object category, as well as a series of taxonomies, each of which hierarchically partitions those same labels (object classes) by a different semantic view. For each taxonomy, we first learn a tree of semantic kernels: each node in a tree has a Mahalanobis-based kernel optimized to distinguish between the classes in its children nodes. The kernels in one tree isolate image features useful at a range of category granularities. Then, using the resulting semantic kernel forest from all taxonomies, we apply a form of multiple kernel learning (MKL) to obtain class-specific kernel combinations, in order to select only those relationships relevant to recognize each object class. We introduce a novel hierarchical regularization term into the MKL objective that further exploits the taxonomies' structure. The output of the method is one learned kernel per object class, which we can then deploy for one-versus-all multi-class classification on novel images.

Our main contribution is to simultaneously exploit multiple semantic taxonomies for visual feature learning. Whereas past work focuses on building object hierarchies for scalable classification [12, 13] or using WordNet to gauge semantic distance [5, 6, 8, 9], we learn discriminative kernels that capitalize on the cues in diverse taxonomy views, leading to better recognition accuracy. The primary technical contributions are i) an approach to generate semantic base kernels across taxonomies, ii) a method to integrate the complementary cues from multiple suboptimal taxonomies, and iii) a novel regularizer for multiple kernel learning that exploits hierarchical structure from the taxonomy, allowing kernel selection to benefit from semantic knowledge of the problem domain.

We demonstrate our approach with challenging images from the Animals with Attributes and ImageNet datasets [14, 7] together with taxonomies spanning cognitive synsets, visual attributes, behavior, and habitats. Our results show that the taxonomies can indeed boost feature learning, letting us benefit from humans' perceived distinctions as implicitly embedded in the trees. Furthermore, we show that interleaving the forest of multiple taxonomic views leads to the best performance, particularly when coupled with the proposed novel regularization.

## 2 Related Work

**Leveraging hierarchies for object recognition**   Most work in object recognition that leverages category hierarchy does so for the sake of efficient classification [15, 16, 12, 13, 17]. Making coarse to fine predictions along a tree of classifiers efficiently rules out unlikely classes at an early stage. Since taxonomies need not be ideal structures for this goal, recent work focuses on novel ways to optimize the tree structure itself [12, 13, 17], while others consider splits based on initial inter-class confusions [16]. A parallel line of work explores unsupervised discovery of hierarchies for image organization and browsing, from images alone [18, 19] or from images and tags [20]. Whereas all such work exploits tree structures to improve efficiency (whether in classification or browsing), our goal is for externally defined semantic hierarchies to enhance recognition accuracy.

More related to our problem setting are techniques that exploit the inter-class relationships in a taxonomy [5, 6, 8, 9, 10, 11]. One idea is to combine the decisions of classifiers along the semantic hierarchy [5, 4]. Alternatively, the semantic "distance" between nodes can be used to penalize misclassifications more meaningfully [9], or to share labeled exemplars between similar classes [8]. Metric learning and feature selection can also benefit from an object hierarchy, either by preferring to use disjoint feature sets to discriminate super- and sub-classes [10], by using a taxonomy-induced loss for structured sparsity [21], or by sharing parameters between metrics along the same path [11]. All prior work commits to a single taxonomy, however, which as discussed above may restrict the semantics' impact and will not always align well with the visual data.

**Classification with multiple semantic views**   Combining information from multiple "views" of data is a well-researched topic in the machine learning, multimedia, and computer vision communities. In *multi-view learning*, the training data typically consists of paired examples coming from different modalities—e.g., text and images, or speech and video; basic approaches include recovering the underlying shared latent space for both views [22, 20], bootstrapping classifiers formed independently per feature space [23, 24], or accounting for the view dependencies during clustering [25, 26]. When the classification tasks themselves are grouped, *multi-task learning* methods leverage the parallel tasks to regularize parameters learned for the individual classifiers or features (e.g., [27, 28, 29]). Broadly speaking, our problem has a similar spirit to such settings, since we want to leverage multiple parallel taxonomies over the data; however, our goal to aggregate portions of the taxonomies during feature learning is quite distinct. More specifically, while previous methods attempt to find a single structure to accommodate both views, we seek complementary information from the semantic views and assemble task-specific discriminative features.

**Learning kernel combinations**   Multiple kernel learning (MKL) algorithms [30] have shown promise for image recognition (e.g., [31, 32]) and are frequently employed in practice as a principled way to combine feature types. Our approach also employs a form of MKL, but rather than pool kernels stemming from different low-level features or kernel hyperparameters, it pools kernels stemming from different semantic sources. Furthermore, our addition of a novel regularizer exploits the hierarchical structure from which the kernels originate.

## 3 Approach

We cast the problem of learning semantic features from multiple taxonomies as learning to combine kernels. The base kernels capture features specific to individual taxonomies and granularities within those taxonomies, and they are combined discriminatively to improve classification, weighing each taxonomy and granularity only to the extent useful for the target classification task.

We describe the two main components of the approach in turn: learning the base kernels—which we call a *semantic kernel forest* (Sec. 3.1), and learning their combination across taxonomies (Sec. 3.2), where we devise a new hierarchical regularizer for MKL.

In what follows, we assume that we are given a labeled dataset $\mathcal{D} = \{(\boldsymbol{x}_i, y_i)\}_{n=1}^{\mathsf{N}}$ where $(\boldsymbol{x}_i, y_i)$ stands for the $i$th instance (feature vector) and its class label is $y_i$, as well as a set of tree-structured taxonomies $\{\mathcal{T}_t\}_{t=1}^{\mathsf{T}}$. Each taxonomy $\mathcal{T}_t$ is a collection of nodes. The leaf nodes correspond to class labels, and the inner nodes correspond to superclasses—or, more generally, *semantically meaningful groupings of categories*. We index those nodes with double subscripts $tn$, where $t$ refers to the $t$th taxonomy and $n$ to the $n$th node in that taxonomy. Without loss of generality, we assign the leaf nodes (i.e., the class nodes) a number between 1 and C, where C is the number of class labels.

## 3.1 Learning a semantic kernel forest

Our first step is to learn a forest of base kernels. These kernels are granularity- and view-specific; that is, they are tuned to similarities implied by the given taxonomies. While base kernels are learned *independently* per taxonomy, they are learned *jointly* within each taxonomy, as we describe next.

Formally, for each taxonomy $\mathcal{T}_t$, we learn a set of Gaussian kernels for the superclass at every internal node $tn$ for which $n \geq \mathsf{C} + 1$. The Gaussian kernels are parameterized as

$$K_{tn}(\boldsymbol{x}_i, \boldsymbol{x}_j) = \exp\{-\gamma_{tn} d^2_{\boldsymbol{M}_{tn}}(\boldsymbol{x}_i, \boldsymbol{x}_j)\} = \exp\{-\gamma_{tn}(\boldsymbol{x}_i - \boldsymbol{x}_j)^{\mathrm{T}} \boldsymbol{M}_{tn}(\boldsymbol{x}_i - \boldsymbol{x}_j)\}, \qquad (1)$$

where the Mahalanobis distance metric $\boldsymbol{M}_{tn}$ is used in lieu of the conventional Euclidean metric. Note that for leaf nodes where $n \leq \mathsf{C}$, we do not learn base kernels.

We want the base kernels to encode similarity between examples using features that reflect their respective granularity in the taxonomy. Certainly, the kernel $K_{tn}$ should home in on features that are helpful to distinguish the node $tn$'s subclasses. Beyond that, however, we specifically want it to use features that are *as different as possible* from the features used by its ancestors. Doing so ensures that the subsequent combination step can choose a sparse set of "disconnected" features.

To that end, we apply our Tree of Metrics (ToM) technique [10] to learn the Mahalanobis parameters $\boldsymbol{M}_{tn}$. In ToM, metrics are learned by balancing two forces: i) discriminative power and ii) a preference for different features to be chosen between parent and child nodes. The latter exploits the taxonomy semantics, based on the intuition that features used to distinguish more abstract classes (dog vs. cat) should differ from those used for finer-grained ones (Siamese vs. Persian cat).

Briefly, for each node $tn$, the training data is reduced to $\mathcal{D}_n = \{(\boldsymbol{x}_i, y_{in})\}$, where $y_{in}$ is the label of $n$'s *child* on the path to the leaf node $y_i$. If $y_i$ is not a descendant of the superclass at the node $n$, then $\boldsymbol{x}_i$ is excluded from $\mathcal{D}_n$. The metrics are learned jointly, with each node mutually encouraging the others to use non-overlapping features. ToM achieves this by augmenting a large margin nearest neighbor [33] loss function $\sum_n \ell(\mathcal{D}_n; \mathcal{M}_{tn})$ with the following *disjoint sparsity regularizer*:

$$\Omega_d(\boldsymbol{M}) = \lambda \sum_{n \geq \mathsf{C}+1} \mathsf{Trace}[\boldsymbol{M}_{tn}] + \mu \sum_{n \geq \mathsf{C}+1} \sum_{m \sim n} \|\mathsf{diag}(\boldsymbol{M}_{tn}) + \mathsf{diag}(\boldsymbol{M}_{tm})\|_2^2, \qquad (2)$$

where $m \sim n$ denotes that node $m$ is either an ancestor or descendant of $n$. The first part of the regularizer encourages sparsity in the diagonal elements of $\boldsymbol{M}_{tn}$, and the second part incurs a penalty when two different metrics "compete" for the same diagonal element, i.e., to use the same feature dimension. The resulting optimization problem is convex and can be solved efficiently [10].

After learning the metrics $\{\boldsymbol{M}_{tn}\}$ in each taxonomy, we construct base kernels as in eq. (1). The bandwidths $\gamma_{tn}$ are set as the average distances on training data. We call the collection $\mathcal{F} = \{K_{tn}\}$ of all base kernels the *semantic kernel forest*. Figure 1 shows an illustrative example.

While ToM has shown promising results in learning metrics in a single taxonomy, its reliance on linear Mahalanobis metrics is inherently limited. A straightforward convex combination of ToMs would result in yet another linear mapping, incapable of capturing nonlinear inter-taxonomic interactions. In contrast, our kernel approach retains ToM's granularity-specific features but also enables nontrivial (nonlinear) combinations, especially when coupled with a novel hierarchical regularizer, which we will define next.

## 3.2 Learning class-specific kernels across taxonomies

Base kernels in the semantic kernel forest are learned jointly *within* each taxonomy but independently *across* taxonomies. To leverage multiple taxonomies and to capture different semantic views of the object categories, we next combine them discriminatively to improve classification.

**Basic setting** To learn class-specific features (or kernels), we compose a one-versus-rest supervised learning problem. Additionally, instead of combining all the base kernels in the forest $\mathcal{F}$, we pre-select a subset of them based on the taxonomy structure. Specifically, from each taxonomy, we select base kernels that correspond to the nodes on the path from the root to the leaf node class. For example, in the Biological taxonomy of Figure 1, for the category *Dalmatian*, this path includes the nodes (superclasses) *canine* and *animal*. Thus, for class $c$, the linearly combined kernel is given by

$$F_c(\boldsymbol{x}_i, \boldsymbol{x}_j) = \sum_t \sum_{n \sim c} \beta_{ctn} K_{tn}(\boldsymbol{x}_i, \boldsymbol{x}_j), \qquad (3)$$

where $n \sim c$ indexes the nodes that are ancestors of $c$, which is a leaf node (recall that the first $\mathsf{C}$ nodes in every taxonomy are reserved for leaf class nodes). The combination coefficients $\beta_{ctn}$ are constrained to be nonnegative to ensure the positive semidefiniteness of the resulting kernel $F_c(\cdot, \cdot)$.

We apply the kernel $F_c(\cdot, \cdot)$ to construct the one-versus-rest binary classifier to distinguish instances from class $c$ from all other classes. We then optimize $\boldsymbol{\beta}_c = \{\beta_{ctn}\}$ such that the classifier attains the lowest empirical misclassification risk. The resulting optimization (in its dual formulation) is analogous to standard multiple kernel learning [30]:

$$\min_{\boldsymbol{\beta}_c} \max_{\boldsymbol{\alpha}_c} \quad \sum_i \alpha_{ci} - \frac{1}{2} \sum_i \sum_j \alpha_{ci} \alpha_{cj} q_{ci} q_{cj} F_c(x_i, x_j)$$

$$\text{s.t.} \quad \sum_i \alpha_{ci} q_{ci} = 0, \quad 0 \leq \alpha_{ci} \leq C, \quad \forall \ i, \tag{4}$$

where $\boldsymbol{\alpha}_c$ is the Lagrange multipliers for the binary SVM classifier, $C$ is the regularizer for the SVM's hinge loss function, and $q_{ci} = \pm 1$ is the indicator variable of whether or not $\boldsymbol{x}_i$'s label is $c$.

**Hierarchical regularization**    Next, we extend the basic setting to incorporate richer modeling assumptions. We hypothesize that kernels at higher-level nodes should be preferred to lower-level nodes. Intuitively, higher-level kernels relate to more classes, thus are likely essential to reduce loss.

We leverage this intuition and knowledge about the relative priority of the kernels from each taxonomy's hierarchical structure. We design a novel structural regularization that prefers larger weights for a parent node compared to its children. Formally, the proposed MKL-H regularizer is given by:

$$\Omega(\boldsymbol{\beta}_c) = \lambda \sum_{t, n \sim c} \beta_{ctn} + \mu \sum_{t, n \sim c} \max(0, \beta_{ctn} - \beta_{ctp_n} + 1). \tag{5}$$

The first part prefers a sparse set of kernels. The second part (in the form of hinge loss) encodes our desire to have the weight assigned to a node $n$ be less than the weight assigned to the node's parent $p_n$. We also introduce a margin of 1 to further increase the difference between the two weights.

Hierarchical regularization was previously explored in [34], where a mixed $(1, 2)$-norm is used to regularize the relative sizes between the parent and the children. The main idea there is to discard children nodes if the parent is not selected. Our regularizer is similar, but is simpler and more computationally efficient. (Additionally, our preliminary studies show [34] has no empirical advantage over our approach in improving recognition accuracy.)

### 3.3   Numerical optimization

Our learning problem is cast as a convex optimization that balances the discriminative loss in eq. (4) and the regularizer in eq. (5):

$$\min_{\boldsymbol{\beta}_c} f(\boldsymbol{\beta}_c) = g(\boldsymbol{\beta}_c) + \Omega(\boldsymbol{\beta}_c), \quad \text{s.t.} \quad \boldsymbol{\beta}_c \geq 0, \tag{6}$$

where we use the function $g(\boldsymbol{\beta})$ to encapsulate the inner maximization problem over $\boldsymbol{\alpha}_c$ in eq. (4).

We use the projected subgradient method to solve eq. (6), for its ease of implementation and practical effectiveness [35]. Specifically, at iteration $t$, let $\boldsymbol{\beta}_c^t$ be the current value of $\boldsymbol{\beta}_c$. We compute $f(\boldsymbol{\beta}_c)$'s subgradient $\boldsymbol{s}_t$, then perform the following update,

$$\boldsymbol{\beta}_c^{t+1} \leftarrow \max\left(0, \ \boldsymbol{\beta}_c^t - \alpha_t \boldsymbol{s}_t\right), \tag{7}$$

where the $\max(\ )$ function implements the projection operation such that the update does not fall outside of the feasible region $\boldsymbol{\beta}_c \geq 0$. For step size $\alpha_t$, we use the modified Polyak's step size [36].

## 4   Experiments

We validate our approach on multiple image datasets, and compare to several informative baselines.

### 4.1   Image datasets and taxonomies

We consider two publicly available image collections: Animals with Attributes (AWA) [14] and ImageNet [7][1]. We form two datasets from AWA. The first consists of the four classes shown in

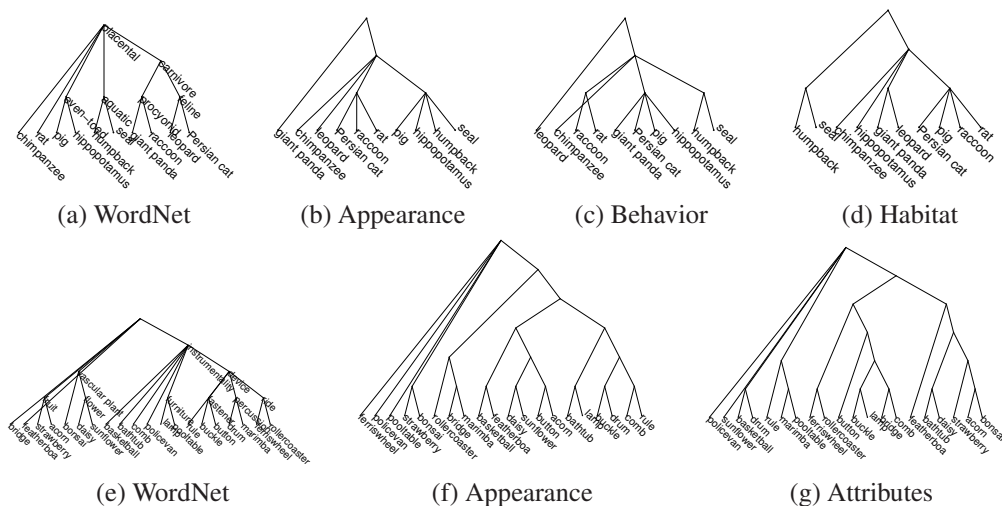

| (a) WordNet | (b) Appearance | (c) Behavior | (d) Habitat |
|---|---|---|---|

| (e) WordNet | (f) Appearance | (g) Attributes |
|---|---|---|

Figure 2: Taxonomies for the AWA-10 (a-d) and ImageNet-20 (e-g) datasets.

Fig. 1, and totals $2,228$ images; the second contains the ten classes in [14], and totals $6,180$ images. We refer to them as **AWA-4** and **AWA-10**, respectively. The third dataset, **ImageNet-20**, consists of $28,957$ total images spanning 20 classes from ILSVRC2010. We chose classes that are non-animals (to avoid overlap with AWA) and that have attribute labels [37].

To obtain multiple taxonomies per dataset, we use attribute labels and WordNet. Attributes are human understandable properties shared among object classes, e.g., "furry", "flat", "carnivorous" [14]. AWA and ImageNet have 85 and 25 attribute labels, respectively. To form semantic taxonomies based on attributes, we first manually divide the attribute labels into subsets according to their mutual semantic relevance (e.g., "furry" and "shiny" are attributes relevant for an Appearance taxonomy, while "land-dwelling" and "aquatic" are relevant for a Habitat taxonomy). Then, for each subset of attributes, we perform agglomerative clustering using Euclidean distance on vectors of the training images' real-valued attributes. We restrict the tree height (6 for ImageNet and 3 for AWA) to ensure that the branching factor at the root is not too high. To extract a WordNet taxonomy, we find all nodes in WordNet that contain the object class names on their word lists, and then build a hierarchy by pruning nodes with only one child and resolving multiple parentship.

For AWA-10, we use 4 taxonomies: one from WordNet, and three based on attribute subsets reflecting Appearance, Behavior, and Habitat ties. For ImageNet-20, we use 3 taxonomies: one from WordNet, one reflecting Appearance as found by hierarchical clustering on the visual features, and one reflecting Attributes using annotations from [37]. For the AWA-4 taxonomies, we simply generate all 3 possible 2-level binary trees, which, based on manual observation, yield taxonomies reflecting Biological, Appearance, and Habitat ties between the animals. See Figures 1 and 2.

We stress that these taxonomies are created *externally with human knowledge*, and thus they inject perceived object relationships into the feature learning problem. This is in stark contrast to prior work that focuses on optimizing hierarchies for efficiency, without requiring interpretability of the trees themselves [16, 12, 13, 17].

## 4.2 Baseline methods for comparison

We compare our method to three key baselines: **1) Raw feature kernel:** an RBF kernel computed on the original image features, with the $\gamma$ parameter set to the inverse of the mean Euclidean distance $d$ among training instances. **2) Raw feature kernel + MKL:** MKL combination of multiple such RBF kernels constructed by varying $\gamma$, which is a traditional approach to generate base kernels (e.g., [30]). For this baseline, we generate the same number $N$ of base kernels as in the semantic kernel forest, with $\gamma = \frac{\sigma}{d}$, for $\sigma = \{2^{1-m}, \ldots, 2^{N-m}\}$, where $m = \frac{N}{2}$. **3) Perturbed semantic kernel tree:** a semantic kernel tree trained with taxonomies that have randomly swapped leaves.

|  | AWA-4 | AWA-10 | ImageNet-20 |
|---|---|---|---|
| Raw feature kernel | $47.67 \pm 2.22$ | $30.80 \pm 1.36$ | $28.20 \pm 1.45$ |
| Raw feature kernel + MKL | $48.50 \pm 1.89$ | $31.13 \pm 2.81$ | $27.67 \pm 1.50$ |
| Perturbed semantic kernel tree + MKL-H | N/A | $31.53 \pm 2.07$ | $28.20 \pm 2.02$ |
| Perturbed semantic kernel forest + MKL-H | N/A | $33.20 \pm 2.96$ | $30.77 \pm 1.53$ |
| Semantic kernel tree + Avg | $47.17 \pm 2.40$ | $31.92 \pm 1.21$ | $28.97 \pm 1.61$ |
| Semantic kernel tree + MKL | $48.89 \pm 1.06$ | $32.43 \pm 1.93$ | $29.74 \pm 1.26$ |
| Semantic kernel tree + MKL-H | $50.06 \pm 1.12$ | $32.68 \pm 1.79$ | $29.90 \pm 0.70$ |
| Semantic kernel forest + MKL | $49.67 \pm 1.11$ | $34.60 \pm 1.78$ | $30.97 \pm 1.14$ |
| Semantic kernel forest + MKL-H | $\mathbf{52.83 \pm 1.68}$ | $\mathbf{35.87 \pm 1.22}$ | $\mathbf{32.30 \pm 1.00}$ |

Table 1: Multi-class classification accuracy on all datasets, across 5 train/test splits. (The perturbed semantic kernel tree baseline is not applicable for AWA-4, since all possible groupings are present in the taxonomies.)

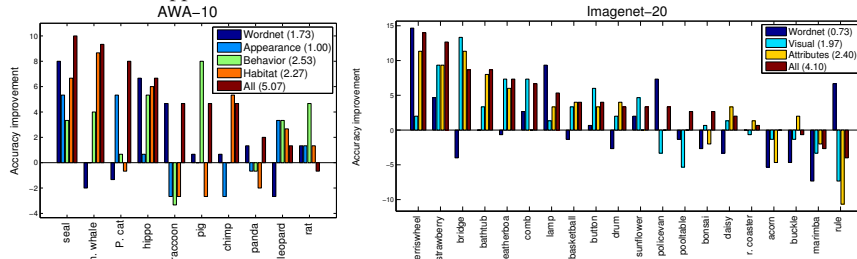

Figure 3: Per-class accuracy improvements of each individual taxonomy and the semantic kernel forest ("All") over the raw feature kernel baseline. Numbers in legends denote mean improvement. Best viewed in color.

The first two baselines will show the accuracy attainable using the same image features and basic classification tools (SVM, MKL) as our approach, but lacking the taxonomy insights. The last baseline will test if weakening the semantics in the taxonomy has a negative impact on accuracy.

We evaluate several variants of our approach, in order to analyze the impact of each component: **1) Semantic kernel tree + Avg:** an equal-weight average of the semantic kernels from one taxonomy. **2) Semantic kernel tree + MKL:** the same kernels, but combined with MKL using sparsity regularization only (i.e., $\mu = 0$ in eq. 5). **3) Semantic kernel tree + MKL-H:** the same as previous, but adding the proposed hierarchical regularization (eq. 5). **4) Semantic kernel forest + MKL:** semantic forest kernels from multiple taxonomies combined with MKL. **5) Semantic kernel forest + MKL-H:** the same as previous, but adding our hierarchical regularizer.

### 4.3 Implementation details

For all results, we use $30/30/30$ images per class for training/validation/testing, and generate 5 such random splits. We report average multi-class recognition accuracy and standard errors for $95\%$ confidence interval. For single taxonomy results, we report the average over all individual taxonomies. For all methods, the raw image features are bag-of-words histograms obtained on SIFT, provided with the datasets. We reduce their dimensionality to 100 with PCA to speed up the ToM training, following [10]. To train ToM, we sample $400$ random constraints and cross-validate the regularization parameters $\lambda, \gamma \in \{0.1, 1, 10\}$. For MKL/MKL-H, we use $C = 1000$ for the C-SVM parameter, and cross-validate the sparsity and hierarchical parameters $\lambda, \mu \in \{0, 0.1, 1, 10\}$.

### 4.4 Results

**Quantitative results**   Table 1 shows the multi-class classification accuracy on all three datasets. Our semantic kernel forests approach significantly outperforms all three baselines. It improves accuracy for 9 of the 10 AWA-10 classes, and 16 of the 20 classes in ImageNet-20 (see Figure 3). These gains clearly show the impact of injecting semantics into discriminative feature learning. The forests' advantage over the individual trees supports our core claim regarding the value of interleaving semantic cues from multiple taxonomies. Further, the proposed hierarchical regularization (MKL-H) outperforms the generic MKL, particularly for the multiple taxonomy forests.

We stress that semantic kernel forests' success is *not* simply due to having access to a variety of kernels, as we can see by comparing our method to both the raw feature MKL and perturbed tree

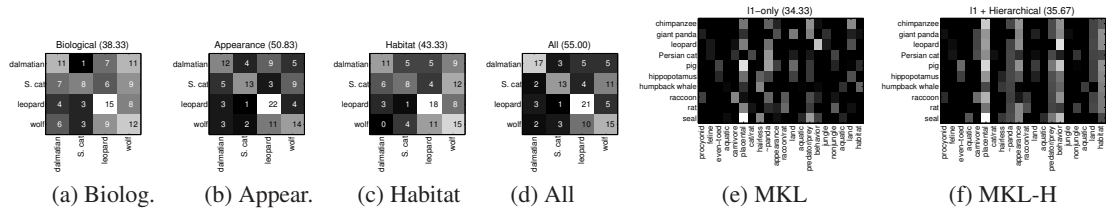

| (a) Biolog. | (b) Appear. | (c) Habitat | (d) All | (e) MKL | (f) MKL-H |

Figure 4: (a-d): AWA-4 confusion matrices for individual taxonomies (a-c) and the combined taxonomies (d). Y-axis shows true classes; x-axis shows predicted classes. (e-f): Example $\beta_c$'s to show the characteristics of the two regularizers. Each entry is a learned kernel weight (brighter=higher weight). Y-axis shows object classes; x-axis shows kernel node names.

results—all of which use the same number of kernels. Instead, the advantage is leveraging the implicit discriminative criteria embedded in the external semantic groupings. In addition, we note that even perturbed taxonomies can be semantic; some of their groupings of classes may happen to be meaningful, especially when there are fewer categories. Hence, their advantage over the raw feature kernels is understandable. Nonetheless, perturbed taxonomies are semantically weaker than the originals, and our kernel trees with the true single or multiple taxonomies perform better.

MKL-H has the most impact for the multiple taxonomy forests, and relatively little on the single kernel tree. This makes sense. For a single taxonomy, a single kernel is solely responsible for discriminating a class from the others, making all kernels similarly useful. In contrast, in the forest, two classes are related at multiple different nodes, making it necessary to select out useful views; here, the hierarchical regularizer plays the role of favoring kernels at higher levels, which might have more generalization power due to the training set size and number of classes involved.

The per-class and per-taxonomy comparisons in Figure 3 further elucidate the advantage of using multiple complementary taxonomies. A single semantic kernel tree often improves accuracy on some classes, but at the expense of reduced accuracy on others. This illustrates that the structure of an individual taxonomy is often suboptimal. For example, the Habitat taxonomy on AWA-10 helps distinguish *humpback whale* well from the others—it branches early from the other animals due to its distinctive "oceanic" background—but it hurts accuracy for *giant panda*. The WordNet taxonomy does exactly the opposite, improving *giant panda* via the Biological taxonomy, but hurting *humpback whale*. The semantic kernel forest takes the best of both through its learned combination. The only cases in which it fails are when the majority of the taxonomies strongly degrade performance, as to be expected given the linear MKL combination (e.g., see the classes *marimba* and *rule*).

**Further qualitative analysis** Figure 4 (a-d) shows the confusion matrices for AWA-4 using only the root level kernels. We see how each taxonomy specializes the features, exactly in the manner sketched in Sec. 1. The combination of all taxonomies achieves the highest accuracy (55.00), better than the maximally performing individual taxonomy (Appearance, 50.83). Figure 4 (e-f) shows the learned kernel combination weights $\beta_c$ for each class $c$ in AWA-10, using the two different regularizers. In (e), the $L1$ regularizer selects a sparse set of useful kernels. For example, the *humpback whale* drops the kernels belonging to the whole Behavior taxonomy block, and gives the strongest weight to "hairless", and "habitat". However, by failing to select some of the upper-level nodes, it focuses only on the most confusing fine-grained problems. In contrast, with the proposed regularization (f), we see more emphasis on the upper nodes (e.g., the "behavior" and "placental" kernels), which helps accuracy.

## 5 Conclusion

We proposed a semantic kernel forest approach to learn discriminative visual features that leverage information from multiple semantic taxonomies. The results show that it improves object recognition accuracy, and give good evidence that committing to a single external knowledge source is insufficient. In future work, we plan to explore non-additive and/or local per-instance kernel combination techniques for integrating the semantic views.

**Acknowledgements** This research is supported in part by NSF IIS-1065243 and NSF IIS-1065390.

## Footnotes

[1]`attributes.kyb.tuebingen.mpg.de/` and `image-net.org/challenges/LSVRC/2011/`

# References

[1] N. Dalal and B. Triggs. Histograms of Oriented Gradients for Human Detection. In *CVPR*, 2005.

[2] J. Wang, J. Yang, K. Yu, F. Lv, T. Huang, and Y. Gong. Locality-Constrained Linear Coding for Image Classification. In *CVPR*, 2010.

[3] C. Fellbaum, editor. *WordNet An Electronic Lexical Database*. MIT Press, May 1998.

[4] A. Zweig and D. Weinshall. Exploiting Object Hierarchy: Combining Models from Different Category Levels. In *ICCV*, 2007.

[5] M. Marszalek and C. Schmid. Semantic hierarchies for visual object recognition. In *CVPR*, 2007.

[6] A. Torralba, R. Fergus, and W. T. Freeman. 80 million Tiny Images: a Large Dataset for Non-Parametric Object and Scene Recognition. *PAMI*, 30(11):1958–1970, 2008.

[7] J. Deng, W. Dong, R. Socher, L.-J. Li, K. Li, and L. Fei-Fei. Imagenet: A Large-Scale Hierarchical Image Database. In *CVPR*, 2009.

[8] R. Fergus, H. Bernal, Y. Weiss, and A. Torralba. Semantic label sharing for learning with many categories. In *ECCV*, 2010.

[9] J. Deng, A. Berg, K. Li, and L. Fei-Fei. What does classifying more than 10,000 image categories tell us? In *ECCV*, 2010.

[10] S. J. Hwang, K. Grauman, and F. Sha. Learning a tree of metrics with disjoint visual features. In *NIPS*, 2011.

[11] N. Verma, D. Mahajan, S. Sellamanickam, and V. Nair. Learning hierarchical similarity metrics. In *CVPR*, 2012.

[12] S. Bengio, J. Weston, and D. Grangier. Label Embedding Trees for Large Multi-Class Task. In *NIPS*, 2010.

[13] J. Deng, S. Satheesh, A. Berg, and L. Fei Fei. Fast and balanced: Efficient label tree learning for large scale object recognition. In *NIPS*, 2011.

[14] C. Lampert, H. Nickisch, and S. Harmeling. Learning to Detect Unseen Object Classes by Between-Class Attribute Transfer. In *CVPR*, 2009.

[15] M. Marszalek and C. Schmid. Constructing category hierarchies for visual recognition. In *ECCV*, 2008.

[16] G. Griffin and P. Perona. Learning and using taxonomies for fast visual categorization. In *CVPR*, 2008.

[17] T. Gao and D. Koller. Discriminative learning of relaxed hierarchy for large-scale visual recognition. In *ICCV*, 2011.

[18] J. Sivic, B. Russell, A. Zisserman, W. Freeman, and A. Efros. Unsupervised discovery of visual object class hierarchies. In *CVPR*, 2008.

[19] E. Bart, I. Porteous, P. Perona, and M. Welling. Unsupervised learning of visual taxonomies. In *CVPR*, 2008.

[20] L.-J. Li, C. Wang, Y. Lim, D. Blei, and L. Fei-Fei. Building and using a semantivisual image hierarchy. In *CVPR*, 2010.

[21] S. Kim and E. Xing. Tree-guided group lasso for multi-task regression with structured sparsity. In *ICML*, 2010.

[22] D. R. Hardoon, S. Szedmak, and J. Shawe-Taylor. Canonical Correlation Analysis: An Overview with Application to Learning Methods. *Neural Computation*, 16(12), 2004.

[23] A. Blum and T. Mitchell. Combining Labeled and Unlabeled Data with Co-training. In *COLT: Proceedings of the Workshop on Computational Learning Theory*, 1998.

[24] C. Christoudias, K. Saenko, L. Morency, and T. Darrell. Co-adaptation of audio-visual speech and gesture classifiers. In *International Conference on Multimodal Interaction*, 2006.

[25] I. Dhillon, S. Mallela, and R. Kumar. A divisive information-theoretic feature clustering algorithm for text classification. *Journal of Machine Learning Research*, 3:1265–1287, 2003.

[26] A. Gupta and S. Dasgupta. Hybrid hierarchical clustering: Forming a tree from multiple views. In *Workshop on Learning With Multiple Views*, 2005.

[27] A. Argyriou, T. Evgeniou, and M. Pontil. Multi-task feature learning. In *NIPS*, 2006.

[28] N. Loeff and A. Farhadi. Scene Discovery by Matrix Factorization. In *ECCV*, 2008.

[29] S. J. Hwang, F. Sha, and K. Grauman. Sharing features between objects and their attributes. In *CVPR*, 2011.

[30] F. Bach, G. Lanckriet, and M. Jordan. Multiple Kernel Learning, Conic Duality, and the SMO Algorithm. In *ICML*, 2004.

[31] M. Varma and D. Ray. Learning the discriminative power-invariance trade-off. In *ICCV*, 2007.

[32] P. Gehler and S. Nowozin. On feature combination for multiclass object classification. In *ICCV*, 2009.

[33] K. Weinberger, J. Blitzer, and L. Saul. Distance Metric Learning for Large Margin Nearest Neighbor Classification. In *NIPS*, 2006.

[34] F. Bach. Exploring large feature spaces with hierarchical multiple kernel learning. In *NIPS*, 2008.

[35] D. Bertsekas. *Nonlinear Programming*. Athena Scientific, 1999.

[36] S. Boyd and A. Mutapcic. Subgradient methods. 2007.

[37] O. Russakovsky and L. Fei-Fei. Attribute learning in large-scale datasets. In *ECCV*, 2010.

